# Dynamics of Generalization in Linear Perceptrons

**Anders Krogh**
Niels Bohr Institute
Blegdamsvej 17
DK-2100 Copenhagen, Denmark

**John A. Hertz**
NORDITA
Blegdamsvej 17
DK-2100 Copenhagen, Denmark

## Abstract

We study the evolution of the generalization ability of a simple linear perceptron with $N$ inputs which learns to imitate a "teacher perceptron". The system is trained on $p = \alpha N$ binary example inputs and the generalization ability measured by testing for agreement with the teacher on all $2^N$ possible binary input patterns. The dynamics may be solved analytically and exhibits a phase transition from imperfect to perfect generalization at $\alpha = 1$. Except at this point the generalization ability approaches its asymptotic value exponentially, with critical slowing down near the transition; the relaxation time is $\propto (1 - \sqrt{\alpha})^{-2}$. Right at the critical point, the approach to perfect generalization follows a power law $\propto t^{-\frac{1}{2}}$. In the presence of noise, the generalization ability is degraded by an amount $\propto (\sqrt{\alpha} - 1)^{-1}$ just above $\alpha = 1$.

## 1 INTRODUCTION

It is very important in practical situations to know how well a neural network will generalize from the examples it is trained on to the entire set of possible inputs. This problem is the focus of a lot of recent and current work [1-11]. All this work, however, deals with the asymptotic state of the network after training. Here we study a very simple model which allows us to follow the evolution of the generalization ability in time under training. It has a single linear output unit, and the weights obey adaline learning. Despite its simplicity, it exhibits nontrivial behaviour: a dynamical phase transition at a critical number of training examples, with power-law decay right at the transition point and critical slowing down as one approaches it from either side.

## 2   THE MODEL

Our simple linear neuron has an output $V = N^{-\frac{1}{2}} \sum_i w_i \xi_i$, where $\xi_i$ is the $i$th input. It learns to imitate a teacher [1] whose weights are $u_i$ by training on $p$ examples of input-output pairs $(\xi_i^\mu, \zeta^\mu)$ with

$$\zeta^\mu = N^{-\frac{1}{2}} \sum_i u_i \xi_i^\mu \tag{1}$$

generated by the teacher. The adaline learning equation [11] is then

$$\dot{w}_i = \frac{1}{\sqrt{N}} \sum_{\mu=1}^{p} (\zeta^\mu - \frac{1}{\sqrt{N}} \sum_j w_j \xi_j^\mu) \xi_i^\mu = \frac{1}{N} \sum_{\mu j} (u_j - w_j) \xi_j^\mu \xi_i^\mu. \tag{2}$$

By introducing the difference between the teacher and the pupil,

$$v_i \equiv u_i - w_i, \tag{3}$$

and the training input correlation matrix

$$A_{ij} = \frac{1}{N} \sum_{\mu=1}^{p} \xi_j^\mu \xi_i^\mu, \tag{4}$$

the learning equation becomes

$$\dot{v}_i = -\sum_j A_{ij} v_j. \tag{5}$$

We let the example inputs $\xi_i^\mu$ take the values $\pm 1$, randomly and independently, but it is straightforward to generalize it to any distribution of inputs with $\langle \xi_i^\mu \xi_j^\nu \rangle_\xi \propto \delta_{ij} \delta_{\mu\nu}$. For a large number of examples ($p = O(N) \gg 1$), the resulting generalization ability will be independent of just which $p$ of the $2^N$ possible binary input patterns we choose. All our results will then depend only on the fact that we can calculate the spectrum of the matrix A.

## 3   GENERALIZATION ABILITY

To measure the generalization ability, we test whether the output of our perceptron with weights $w_i$ agrees with that of the teacher with weights $u_i$ on all possible binary inputs. Our objective function, which we call the generalization error, is just the square of the error, averaged over all these inputs:

$$
\begin{aligned}
F &= \frac{1}{N 2^N} \sum_{\{\sigma\}} \left( \sum_i (u_i - w_i)\sigma_i \right)^2 = \frac{1}{N} \sum_{ij} v_i v_j \frac{1}{2^N} \sum_{\{\sigma\}} \sigma_i \sigma_j \\
&= \frac{1}{N} \sum_i v_i^2
\end{aligned}
\tag{6}
$$

(We used that $\frac{1}{2^N} \sum_{\{\sigma\}} \sigma_i \sigma_j$ is zero unless $i = j$.) That is, $F$ is just proportional to the square of the difference between the teacher and pupil weight vectors. With the

$N^{-1}$ normalization factor $F$ will then vary between 1 (*tabula rasa*) and 0 (perfect generalization) if we normalize $\vec{u}$ to length $\sqrt{N}$. During learning, $w_i$ and thus $v_i$ depends on time, so $F$ is a function of $t$. The complementary quantity $1 - F(t)$ could be called the generalization ability.

In the basis where A is diagonal, the learning equation (5) is simply

$$\dot{v}_r = -A_r v_r \qquad (7)$$

where $A_r$ are the eigenvalues of A. This has the solution

$$v_r(t) = v_r(0)e^{-A_r t} = u_r(0)e^{-A_r t}, \qquad (8)$$

where it is assumed that the weights are zero at time $t = 0$ (we will come back to the more general case later). Thus we find

$$F(t) = \frac{1}{N} \sum_r v_r^2(t) = \frac{1}{N} \sum_r u_r^2 e^{-2A_r t} \qquad (9)$$

Averaging over all possible training sets of size $p$ this can be expressed in terms of the density of eigenvalues of A, $\rho(\epsilon)$:

$$F(t) = \frac{|\vec{u}|^2}{N} \int d\epsilon \rho(\epsilon) e^{-2\epsilon t}. \qquad (10)$$

In the following it will be assumed that the length of $\vec{u}$ is normalized to $\sqrt{N}$, so the prefactor disappears.

For large $N$, the eigenvalue density is (see, e.g. [11], where it can be obtained simply from the imaginary part of the Green's function in eq.(57))

$$\rho(\epsilon) = \frac{1}{2\pi\epsilon} \sqrt{(\epsilon_+ - \epsilon)(\epsilon - \epsilon_-)} + (1 - \alpha)\theta(1 - \alpha)\delta(\epsilon), \qquad (11)$$

where

$$\epsilon_\pm = (1 \pm \sqrt{\alpha})^2 \qquad (12)$$

and $\theta(\ )$ is the unit step function. The density has two terms: a 'deformed semicircle' between the roots $\epsilon_-$ and $\epsilon_+$, and for $\alpha < 1$ a delta function at $\epsilon = 0$ with weight $1 - \alpha$. The delta-function term appears because no learning takes place in the subspace orthogonal to that spanned by the training patterns. For $\alpha > 1$ the patterns span the whole space, and therefore the delta-function is absent.

The results at infinite time are immediately evident. For $\alpha < 1$ there is a nonzero limit, $F(\infty) = 1 - \alpha$, while $F(\infty)$ vanishes for $\alpha \geq 1$, indicating perfect generalization (the solid line in Figure 1). While on the one hand it may seem remarkable that perfect generalization can be obtained from a training set which forms an infinitesimal fraction of the entire set of possible examples, the meaning of the result is just that $N$ points are sufficient to determine an $N - 1$-dimensional hyperplane in $N$ dimensions.

Figure 2 shows $F(t)$ as obtained numerically from (10) and (11). The qualitative form of the approach to $F(\infty)$ can be obtained analytically by inspection. For $\alpha \neq 1$, the asymptotic approach is governed by the smallest nonzero eigenvalue $\epsilon_-$. Thus we have critical slowing down, with a divergent relaxation time

$$\tau = \frac{1}{\epsilon_-} = \frac{1}{|\sqrt{\alpha} - 1|^2} \qquad (13)$$

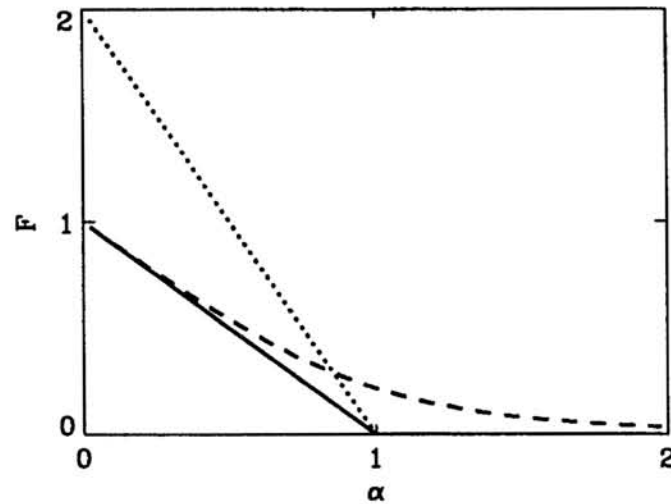

Figure 1: The asymptotic generalization error as a function of $\alpha$. The full line corresponds to $\lambda = 0$, the dashed line to $\lambda = 0.2$, and the dotted line to $w_0 = 1$ and $\lambda = 0$.

as the transition at $\alpha = 1$ is approached. Right at the critical point, the eigenvalue density diverges for small $\epsilon$ like $\epsilon^{-\frac{1}{2}}$, which leads to the power law

$$F(t) \propto \frac{1}{\sqrt{t}} \tag{14}$$

at long times. Thus, while exactly $N$ examples are sufficient to produce perfect generalization, the approach to this desirable state is rather slow. A little bit above $\alpha = 1$, $F(t)$ will also follow this power law for times $t \ll \tau$, going over to (slow) exponential decay at very long times ($t \gg \tau$). By increasing the training set size well above $N$ (say, to $\frac{3}{2}N$), one can achieve exponentially fast generalization. Below $\alpha = 1$, where perfect generalization is never achieved, there is at least the consolation that the approach to the generalization level the network does reach is exponential (though with the same problem of a long relaxation time just below the transition as just above it).

## 4   EXTENSIONS

In this section we briefly discuss some extensions of the foregoing calculation. We will see what happens if the weights are non-zero at $t = 0$, discuss weight decay, and finally consider noise in the learning process.

Weight decay is a simple and frequently-used way to limit the growth of the weights, which might be desirable for several reasons. It is also possible to approximate the problem with binary weights using a weight decay term (the so-called spherical model, see [11]). We consider the simplest kind of weight decay, which comes in as an additive term, $-\lambda w_i = -\lambda(u_i - v_i)$, in the learning equation (2), so the equation

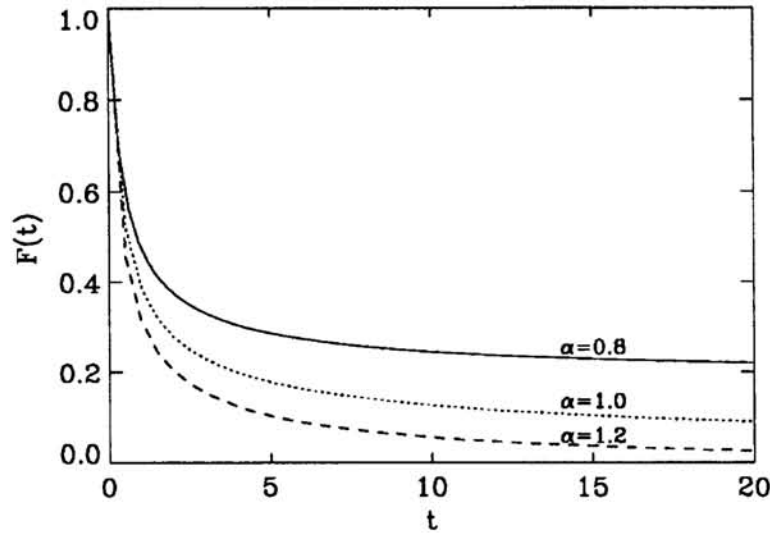

Figure 2: The generalization error as a function of time for a couple of different $\alpha$.

(5) for the difference between teacher and pupil is now

$$\dot{v}_i = -\sum_j A_{ij}v_j + \lambda(u_i - v_i) = -\sum_j (A_{ij} + \lambda\delta_{ij})v_j + \lambda u_i. \tag{15}$$

Apart from the last term this just shifts the eigenvalue spectrum by $\lambda$.

In the basis where A is diagonal we can again write down the general solution to this equation:

$$v_r = \frac{(1 - e^{-(A_r+\lambda)t})\lambda u_r}{A_r + \lambda} + v_r(0)e^{-(A_r+\lambda)t}. \tag{16}$$

The square of this is

$$v_r^2 = u_r^2 \left[ \frac{\lambda(1 - e^{-(A_r+\lambda)t})}{A_r + \lambda} + e^{-(A_r+\lambda)t} + \frac{w_r(0)}{u_r}e^{-(A_r+\lambda)t} \right]^2. \tag{17}$$

As in (10) this has to be integrated over the eigenvalue spectrum to find the averaged generalization error. Assuming that the initial weights are random, so that $\overline{w_r(0)} = 0$, and that they have a relative variance given by

$$\overline{\left(\frac{w_r(0)}{u_r}\right)^2} = w_0^2, \tag{18}$$

the average of $F(t)$ over the distibution of initial conditions now becomes

$$F(t) = \int d\epsilon \rho(\epsilon) \left[ \left( \frac{\lambda(1 - e^{-(\epsilon+\lambda)t})}{\epsilon + \lambda} + e^{-(\epsilon+\lambda)t} \right)^2 + w_0^2 e^{-2(\epsilon+\lambda)t} \right]. \tag{19}$$

(Again it is assumed the length of $\vec{u}$ is $\sqrt{N}$.)

For $\lambda = 0$ we see the result is the same as before except for a factor $1 + w_0^2$ in front of the integral. This means that the asymptotic generalization error is now

$$F(\infty) = \begin{cases} (1 + w_0^2)(1 - \alpha) & \text{for } \alpha \leq 1 \\ 0 & \text{for } \alpha > 1, \end{cases} \tag{20}$$

which is shown as a dotted line in Figure 1 for $w_0 = 1$. The excess error can easily be understood as a contribution to the error from the non-relaxing part of the initial weight vector in the subspace orthogonal to the space spanned by the patterns. The relaxation times are unchanged for $\lambda = 0$.

For $\lambda > 0$ the relaxation times become finite even at $\alpha = 0$, because the smallest eigenvalue is shifted by $\lambda$, so (13) is now

$$\tau = \frac{1}{\epsilon_- + \lambda} = \frac{1}{|\sqrt{\alpha} - 1|^2 + \lambda}. \tag{21}$$

In this case the asymptotic error can easily be obtained numerically from (19), and is shown by the dashed line in Figure 1. It is *smaller* than for $\lambda = 0$ for $w_0^2 > 1$ at sufficiently small $\alpha$. This is simply because the weight decay makes the part of $\vec{w}(0)$ orthogonal to the pattern space decay away exponentially, thereby eliminating the excess error due to large initial weight components in this subspace.

This phase transition is very sensitive to noise. Consider adding a noise term $\eta_i(t)$ to the right-hand side of (2), with

$$\langle \eta_i(t)\eta_j(t') \rangle = 2T\delta(t - t'). \tag{22}$$

Here we restrict our attention to the case $\lambda = 0$. Carrying the extra term through the succeeding manipulations leads, in place of (7), to

$$\dot{v}_r = -A_r v_r + \eta_r(t). \tag{23}$$

The additional term leads to a correction (after Fourier transforming)

$$\delta v_r(\omega) = \frac{\eta_r(\omega)}{-i\omega + A_r} \tag{24}$$

and thus to an extra (time-independent) piece of the generalization error $F(t)$:

$$\delta F = \frac{1}{N} \sum_r \int \frac{d\omega}{2\pi} \frac{\langle |\eta_r(\omega)|^2 \rangle}{|-i\omega + A_r|^2} = \frac{1}{N} \sum_r \frac{T}{A_r}. \tag{25}$$

For $\alpha > 1$, where there are no zero eigenvalues, we have

$$\delta F = T \int_{\epsilon_-}^{\epsilon_+} d\epsilon \frac{\rho(\epsilon)}{\epsilon} \tag{26}$$

which has the large $\alpha$-limit $T/\alpha$, as found in equilibrium analyses (also for threshold perceptrons [2,3,5,6,7,8,9]). Equation (26) gives a generalization error which diverges as one approaches the transition at $\alpha = 1$:

$$\delta F \propto T\epsilon_-^{-1/2} = \frac{T}{\sqrt{\alpha} - 1}. \tag{27}$$

Equation (25) blows up for $\alpha < 1$, where some of the $A_r$ are zero. This divergence just reflects the fact that in the subspace orthogonal to the training patterns, $\vec{v}$ feels only the noise and so exhibits a random walk whose variance diverges as $t \to \infty$. Keeping more careful track of the dynamics in this subspace leads to

$$\delta F \;=\; 2T(1-\alpha)t + T \int_{\epsilon_-}^{\epsilon_+} d\epsilon \frac{\rho(\epsilon)}{\epsilon}$$

$$\xrightarrow[\alpha \to 1^-]{} \; 2T\left[(1-\alpha)t + O(\tfrac{1}{1-\sqrt{\alpha}})\right] \tag{28}$$

# 5  CONCLUSION

Generalization in the linear perceptron can be understood in the following picture. To get perfect generalization the training pattern vectors have to span the whole input space — $N$ points (in general position) are enough to specify any hyperplane. This means that perfect generalization appears only for $\alpha \geq 1$. As $\alpha$ approaches 1 the relaxation time – i.e. learning time – diverges, signaling a phase transition, as is common in physical systems. Noise has a severe effect on this transition. It leads to a degradation of the generalization ability which diverges as one reduces the number of training examples toward the critical number.

This model is of course much simpler than most real-life training problems. However, it does allow us to examine in detail the dynamical phase transition separating perfect from imperfect generalization. Further extensions of the model can also be solved and will be reported elsewhere.

# References

[1] Gardner, E. and B. Derrida: Three Unfinished Works on the Optimal Storage Capacity of Networks. *Journal of Physics A* **22**, 1983–1994 (1989).

[2] Schwartz, D.B., V.K. Samalam, S.A. Solla, and J.S. Denker: Exhaustive Learning. *Neural Computation* **2**, 371–382 (1990).

[3] Tishby, N., E. Levin, and S.A. Solla: Consistent Inference of Probabilities in Layered Networks: Predictions and Generalization. Proc. IJCNN Washington 1989, vol. 2 403–410, Hillsdale: Erlbaum (1989).

[4] Baum, E.B. and D. Haussler: What Size Net Gives Valid Generalization. *Neural Computation* **1**, 151–160 (1989).

[5] Györgyi, G. and N. Tishby: Statistical Theory of Learning a Rule. In *Neural Networks and Spin Glasses*, eds W.K. Theumann and R. Koeberle. Singapore: World Scientific (1990).

[6] Hansel, D. and H. Sompolinsky: Learning from Examples in a Single-Layer Neural Network. *Europhysics Letters* **11**, 687–692 (1990).

[7] Vallet, F., J. Cailton and P. Refregier: Linear and Nonlinear Extension of the Pseudo-Inverse Solution for Learning Boolean Functions. *Europhysics Letters* **9**, 315-320 (1989).

[8] Opper, M., W. Kinzel, J. Kleinz, and R. Nehl: On the Ability of the Optimal Perceptron to Generalize. *Journal of Physics A* **23**, L581–L586 (1990).

[9] Levin, E., N. Tishby, and S. A. Solla: A Statistical Approach to Learning and Generalization in Layered Neural Networks. AT&T Bell Labs, preprint (1990).

[10] Györgyi, G.: Inference of a Rule by a Neural Network with Thermal Noise. *Physical Review Letters* **64**, 2957–2960 (1990).

[11] Hertz, J.A., A. Krogh, and G.I. Thorbergsson: Phase Transitions in Simple Learning. *Journal of Physics A* **22**, 2133–2150 (1989).
